# Neural Network Routing for Random Multistage Interconnection Networks

**Mark W. Goudreau**
Princeton University
and
NEC Research Institute, Inc.
4 Independence Way
Princeton, NJ 08540

**C. Lee Giles**
NEC Research Institute, Inc.
4 Independence Way
Princeton, NJ 08540

## Abstract

A routing scheme that uses a neural network has been developed that can aid in establishing point-to-point communication routes through multistage interconnection networks (MINs). The neural network is a network of the type that was examined by Hopfield (Hopfield, 1984 and 1985). In this work, the problem of establishing routes through random MINs (RMINs) in a shared-memory, distributed computing system is addressed. The performance of the neural network routing scheme is compared to two more traditional approaches - exhaustive search routing and greedy routing. The results suggest that a neural network router may be competitive for certain RMINs.

## 1 INTRODUCTION

A neural network has been developed that can aid in establishing point-to-point communication routes through multistage interconnection networks (MINs) (Goudreau and Giles, 1991). Such interconnection networks have been widely studied (Huang, 1984; Siegel, 1990). The routing problem is of great interest due to its broad applicability. Although the neural network routing scheme can accommodate many types of communication systems, this work concentrates on its use in a shared-memory, distributed computing system.

Neural networks have sometimes been used to solve certain interconnection network

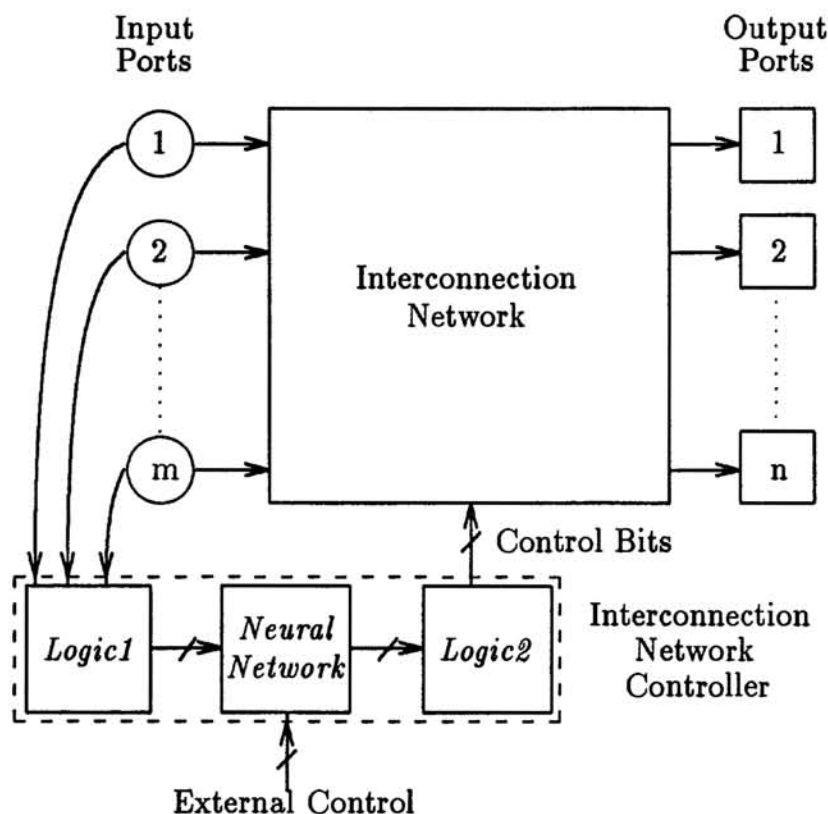

Figure 1: The communication system with a neural network router. The input ports (processors) are on the left, while the output ports (memory modules) are on the right.

problems, such as finding legal routes (Brown, 1989; Hakim and Meadows, 1990) and increasing the throughput of an interconnection network (Brown and Liu, 1990; Marrakchi and Troudet, 1989). The neural network router that is the subject of this work, however, differs significantly from these other routers and is specially designed to handle parallel processing systems that have MINs with random interstage connections. Such random MINs are called RMINs. RMINs tend to have greater fault-tolerance than regular MINs.

The problem is to allow a set of processors to access a set of memory modules through the RMIN. A picture of the communication system with the neural network router is shown in Figure 1. The are $m$ processors and $n$ memory modules. The system is assumed to be synchronous. At the beginning of a message cycle, some set of processors may desire to access some set of memory modules. It is the job of the router to establish as many of these desired connections as possible in a non-conflicting manner. Obtaining the optimal solution is not critical. Stymied processors may attempt communication again during the subsequent message cycle. It is the combination of speed and the quality of the solution that is important.

The object of this work was to discover if the neural network router could be competitive with other types of routers in terms of *quality of solution*, *speed*, and *resource*

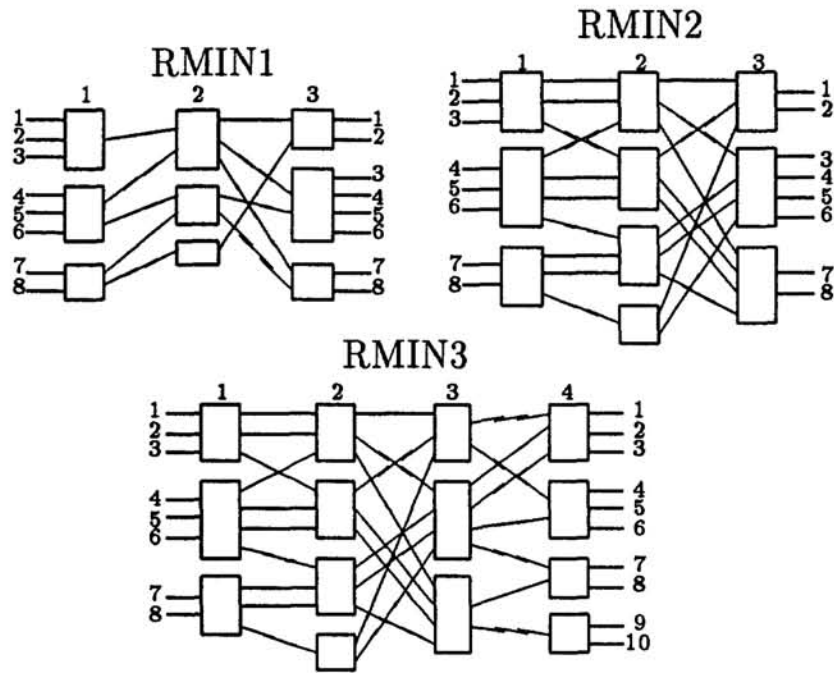

Figure 2: Three random multistage interconnection networks. The blocks that are shown are crossbar switches, for which each input may be connected to each output.

*utilization.* To this end, the neural network routing scheme was compared to two other schemes for routing in RMINs - namely, exhaustive search routing and greedy routing. So far, the results of this investigation suggest that the neural network router may indeed be a practicable alternative for routing in RMINs that are not too large.

## 2    EXHAUSTIVE SEARCH ROUTING

The exhaustive search routing method is optimal in terms of the ability of the router to find the best solution. There are many ways to implement such a router. One approach is described here.

For a given interconnection network, every route from each input to each output was stored in a database. (The RMINs that were used as test cases in this paper always had at least one route from each processor to each memory module.) When a new message cycle began and a new message set was presented to the router, the router would search through the database for a combination of routes for the message set that had no conflicts. A conflict was said to occur if more than one route in the set of routes used a single bus in the interconnection network. In the case where every combination of routes for the message set had a conflict, the router would find a combination of routes that could establish the largest possible number of desired connections.

If there are $k$ possible routes for each message, this algorithm needs a memory of size $\Theta(mnk)$ and, in the worst case, takes exponential time with respect to the size

of the message set. Consequently, it is an impractical approach for most RMINs, but it provides a convenient upper bound for the performance of other routers.

## 3   GREEDY ROUTING

When greedy routing is applied, message connections are established one at a time. Once a route is established in a given message cycle, it may not be removed. Greedy routing does not always provide the optimal routing solution.

The greedy routing algorithm that was used required the same route database as the exhaustive search router did. However, it selects a combination of routes in the following manner. When a new message set is present, the router chooses one desired message and looks at the first route on that message's list of routes. The router then establishes that route. Next, the router examines a second message (assuming a second desired message was requested) and sees if one of the routes in the second message's route list can be established without conflicting with the already established first message. If such a route does exist, the router establishes that route and moves on to the next desired message.

In the worst case, the speed of the greedy router is quadratic with respect to the size of the message set.

## 4   NEURAL NETWORK ROUTING

The focal point of the neural network router is a neural network of the type that was examined by Hopfield (Hopfield, 1984 and 1985). The problem of establishing a set of non-conflicting routes can be reduced to a constraint satisfaction problem. The structure of the neural network router is completely determined by the RMIN. When a new set of routes is desired, only certain bias currents in the network change. The neural network routing scheme also has certain fault-tolerant properties that will not be described here.

The neural network calculates the routes by converging to a legal *routing array*. A legal routing array is 3-dimensional. Therefore, each element of the routing array will have three indices. If element $a_{i,j,k}$ is equal to 1 then message $i$ is routed through output port $k$ of stage $j$. We say $a_{i,j,k}$ and $a_{l,m,n}$ are in the same *row* if $i = l$ and $k = n$. They are in the same *column* if $i = l$ and $j = m$. Finally, they are in the same *rod* if $j = m$ and $k = n$.

A legal routing array will satisfy the following three constraints:

1. one and only one element in each column is equal to 1.
2. the elements in successive columns that are equal to 1 represent output ports that can be connected in the interconnection network.
3. no more than one element in each rod is equal to 1.

The first restriction ensures that each message will be routed through one and only one output port at each stage of the interconnection network. The second restriction ensures that each message will be routed through a legal path in the

interconnection network. The third restriction ensures that any resource contention in the interconnection network is resolved. In other words, only one message can use a certain output port at a certain stage in the interconnection network. When all three of these constraints are met, the routing array will provide a legal route for each message in the message set.

Like the routing array, the neural network router will naturally have a 3-dimensional structure. Each $a_{i,j,k}$ of a routing array is represented by the output voltage of a neuron, $V_{i,j,k}$. At the beginning of a message cycle, the neurons have a random output voltage. If the neural network settles in one of the global minima, the problem will have been solved.

A continuous time mode network was chosen. It was simulated digitally. The neural network has $N$ neurons. The input to neuron $i$ is $u_i$, its input bias current is $I_i$, and its output is $V_i$. The input $u_i$ is converted to the output $V_i$ by a sigmoid function, $g(x)$. Neuron $i$ influences neuron $j$ by a connection represented by $T_{ji}$. Similarly, neuron $j$ affects neuron $i$ through connection $T_{ij}$. In order for the Liapunov function (Equation 5) to be constructed, $T_{ij}$ must equal $T_{ji}$. We further assume that $T_{ii} = 0$. For the synchronous updating model, there is also a time constant, denoted by $\tau$.

The equations which describe the output of a neuron $i$ are:

$$\frac{du_i}{dt} = -\frac{u_i}{\tau} + \sum_{j=1}^{N} T_{ij} V_j + I_i \tag{1}$$

$$\tau = RC \tag{2}$$

$$V_j = g(u_j) \tag{3}$$

$$g(x) = \frac{1}{1 + e^{-x}} \tag{4}$$

The equations above force the neural net into stable states that are the local minima of this approximate energy equation

$$E = -\frac{1}{2} \sum_{i=1}^{N} \sum_{j=1}^{N} T_{ij} V_i V_j - \sum_{i=1}^{N} V_i I_i \tag{5}$$

For the neural network, the weights ($T_{ij}$'s) are set, as are the bias currents ($I_i$'s). It is the output voltages ($V_i$'s) that vary to to minimize $E$.

Let $M$ be the number of messages in a message set, let $S$ be the number of stages in the RMIN, and let $P$ be the number of ports per stage ($P$ may be a function of the stage number). Below are the energy functions that implement the three constraints discussed above:

$$E_1 = \frac{A}{2} \sum_{m=1}^{M} \sum_{s=1}^{S-1} \sum_{p=1}^{P} V_{m,s,p}(-V_{m,s,p} + \sum_{i=1}^{P} V_{m,s,i}) \tag{6}$$

$$E_2 = \frac{B}{2} \sum_{s=1}^{S-1} \sum_{p=1}^{P} \sum_{m=1}^{M} V_{m,s,p}(-V_{m,s,p} + \sum_{i=1}^{M} V_{i,s,p}) \tag{7}$$

$$E_3 = \frac{C}{2} \sum_{m=1}^{M} \sum_{s=1}^{S-1} \sum_{p=1}^{P} (-2V_{m,s,p} + V_{m,s,p}(-V_{m,s,p} + \sum_{i=1}^{P} V_{m,s,i})) \qquad (8)$$

$$E_4 = D \sum_{m=1}^{M} \left[ \sum_{s=2}^{S-1} \sum_{p=1}^{P} \sum_{i=1}^{P} d(s,p,i)V_{m,s-1,p}V_{m,s,i} \right. \qquad (9)$$

$$\left. + \sum_{j=1}^{P} (d(1,\alpha_m,j)V_{m,1,j} + d(S,j,\beta_m)V_{m,S-1,j}) \right]$$

$A$, $B$, $C$, and $D$ are arbitrary positive constants.[1]  $E_1$ and $E_3$ handle the first constraint in the routing array. $E_4$ deals with the second constraint. $E_2$ ensures the third. From the equation for $E_4$, the function $d(s1,p1,p2)$ represents the "distance" between output port $p1$ from stage $s1 - 1$ and output port $p2$ from stage $s1$. If $p1$ can connect to $p2$ through stage $s1$, then this distance may be set to zero. If $p1$ and $p2$ are not connected through stage $s1$, then the distance may be set to one. Also, $\alpha_m$ is the source address of message $m$, while $\beta_m$ is the destination address of message $m$.

The entire energy function is:

$$E = E_1 + E_2 + E_3 + E_4 \qquad (10)$$

Solving for the connection and bias current values as shown in Equation 5 results in the following equations:

$$T_{(m1,s1,p1),(m2,s2,p2)} = -(A+C)\delta_{m1,m2}\delta_{s1,s2}(1-\delta_{p1,p2}) \qquad (11)$$
$$-B\delta_{s1,s2}\delta_{p1,p2}(1-\delta_{m1,m2})$$
$$-D\delta_{m1,m2}[\delta_{s1+1,s2}d(s2,p1,p2) + \delta_{s1,s2+1}d(s1,p2,p1)]$$

$$I_{m,s,p} = C - D[\delta_{s,1}d(1,\alpha_m,p) + \delta_{s,S-1}d(S,p,\beta_m)] \qquad (12)$$

$\delta_{i,j}$ is a Kronecker delta ($\delta_{i,j} = 1$ when $i = j$, and 0 otherwise).

Essentially, this approach is promising because the neural network is acting as a parallel computer. The hope is that the neural network will generate solutions much faster than conventional approaches for routing in RMINs.

The neural network that is used here has the standard problem - namely, a global minimum is not always reached. But this is not a serious difficulty. Typically, when the globally minimal energy is not reached by the neural network, some of the desired routes will have been calculated while others will not have. Even a locally minimal solution may partially solve the routing problem. Consequently, this would seem to be a particularly encouraging type of application for this type of neural network. For this application, the traditional problem of not reaching the global minimum may not hurt the system's performance very much, while the expected speed of the neural network in calculating the solution will be a great asset.

Table 1: Routing results for the RMINs shown in Figure 2. The * entries were not calculated due to their computational complexity.

| | RMIN1 | | | RMIN2 | | | RMIN3 | | |
|---|---|---|---|---|---|---|---|---|---|
| $M$ | $E_{es}$ | $E_{gr}$ | $E_{nn}$ | $E_{es}$ | $E_{gr}$ | $E_{nn}$ | $E_{es}$ | $E_{gr}$ | $E_{nn}$ |
| 1 | 1.00 | 1.00 | 1.00 | 1.00 | 1.00 | 1.00 | 1.00 | 1.00 | 1.00 |
| 2 | 1.86 | 1.83 | 1.87 | 1.97 | 1.97 | 1.98 | 1.99 | 1.88 | 1.94 |
| 3 | 2.54 | 2.48 | 2.51 | 2.91 | 2.91 | 2.93 | 2.99 | 2.71 | 2.87 |
| 4 | 3.08 | 2.98 | 2.98 | 3.80 | 3.79 | 3.80 | 3.94 | 3.49 | 3.72 |
| 5 | 3.53 | 3.38 | 3.24 | 4.65 | 4.62 | 4.61 | * | 4.22 | 4.54 |
| 6 | 3.89 | 3.67 | 3.45 | 5.44 | 5.39 | 5.36 | * | 4.90 | 5.23 |
| 7 | 4.16 | 3.91 | 3.66 | 6.17 | 6.13 | 6.13 | * | 5.52 | 5.80 |
| 8 | 4.33 | 4.10 | 3.78 | 6.86 | 6.82 | 6.80 | * | 6.10 | 6.06 |

The neural network router uses a large number of neurons. If there are $m$ input ports, and $m$ output ports for each stage of the RMIN, an upper bound on the number of neurons needed is $m^2 S$. Often, however, the number of neurons actually required is much smaller than this upper bound.

It has been shown empirically that neural networks of the type used here can converge to a solution in essentially constant time. For example, this claim is made for the neural network described in (Takefuji and Lee, 1991), which is a slight variation of the model used here.

## 5   SIMULATION RESULTS

Figure 2 shows three RMINs that were examined. The routing results for the three routing schemes are shown in Table 1. $E_{es}$ represents the expected number of messages to be routed using exhaustive search routing. $E_{gr}$ is for greedy routing while $E_{nn}$ is for neural network routing. These values are functions of the size of the message set, $M$. Only message sets that did not have obvious conflicts were examined. For example, no message set could have two processors trying to communicate to the same memory module. The table shows that, for at least these three RMINs, the three routing schemes produce solutions that are of similar virtue.

In some cases, the neural network router appears to outperform the supposedly optimal exhaustive search router. That is because the $E_{es}$ and $E_{gr}$ values were calculated by testing *every* message set of size $M$, while $E_{nn}$ was calculated by testing *1,000 randomly generated message sets of size M*. For the neural network router to appear to perform best, it must have gotten message sets that were easier to route than average.

In general, the performance of the neural network router degenerates as the size of the RMIN increases. It is felt that the neural network router in its present form will not scale well for large RMINs. This is because other work has shown that large neural networks of the type used here have difficulty converging to a valid solution (Hopfield, 1985).

# 6   CONCLUSIONS

The results show that there is not much difference, in terms of quality of solution, for the three routing methodologies working on these relatively small sample RMINs. The exhaustive search approach is clearly not a practical approach since it is too time consuming. But when considering the asymptotic analyses for these three methodologies one should keep in mind the performance degradation of the greedy router and the neural network router as the size of the RMIN increases.

Greedy routing and neural network routing would appear to be valid approaches for RMINs of moderate size. But since asymptotic analysis has a very limited significance here, the best way to compare the speeds of these two routing schemes would be to build actual implementations.

Since the neural network router essentially calculates the routes *in parallel*, it can reasonably be hoped that a fast, analog implementation for the neural network router may find solutions faster than the exhaustive search router and even the greedy router. Thus, the neural network router may be a viable alternative for RMINs that are not too large.

## Footnotes

[1] For the simulations, $\tau = 1.0$, $A = C = D = 3.0$, and $B = 6.0$. These values for $A$, $B$, $C$, and $D$ were chosen empirically.

## References

Brown, T. X., (1989), "Neural networks for switching," *IEEE Commun. Mag.*, Vol. 27, pp. 72-81, Nov. 1989.

Brown, T. X. and Liu, K. H., (1990), "Neural network design of a banyan network controller," *IEEE J. on Selected Areas of Comm.*, pp. 1428-1438, Oct. 1990.

Goudreau, M. W. and Giles, C. L., (1991), "Neural network routing for multiple stage interconnection networks," *Proc. IJCNN 91*, Vol. II, p. A-885, July 1991.

Hakim, N. Z. and Meadows, H. E., (1990), "A neural network approach to the setup of the Benes switch," in *Infocom 90*, pp. 397-402.

Hopfield, J. J., (1984), "Neurons with graded response have collective computational properties like those of two-state neurons," *Proc. Natl. Acad. Sci. USA*, Vol. 81, pp. 3088-3092, May 1984.

Hopfield, J. J., (1985), "Neural computation on decisions in optimization problems," *Biol. Cybern.*, Vol. 52, pp. 141-152, 1985.

Huang, K. and Briggs, F. A., (1984), *Computer Architecture and Parallel Processing*, McGraw-Hill, New York, 1984.

Marrakchi, A. M. and Troudet, T., (1989), "A neural net arbitrator for large cross-bar packet-switches," *IEEE Trans. on Circ. and Sys.*, Vol. 36, pp. 1039-1041, July 1989.

Siegel, H. J., (1990), *Interconnection Networks for Large Scale Parallel Processing*, McGraw-Hill, New York, 1990.

Takefuji, Y. and Lee, K. C., (1991), "An artificial hysteresis binary neuron: a model suppressing the oscillatory behaviors of neural dynamics", *Biological Cybernetics*, Vol. 64, pp. 353-356, 1991.
